# Inferring Stimulus Selectivity from the Spatial Structure of Neural Network Dynamics

**Kanaka Rajan**
Lewis-Sigler Institute for Integrative Genomics
Carl Icahn Laboratories # 262, Princeton University
Princeton NJ 08544 USA
krajan@princeton.edu

**L. F. Abbott**
Department of Neuroscience
Department of Physiology and Cellular Biophysics
Columbia University College of Physicians and Surgeons
New York, NY 10032-2695 USA
lfa2103@columbia.edu

**Haim Sompolinsky**
Racah Institute of Physics
Interdisciplinary Center for Neural Computation
Hebrew University
Jerusalem, Israel
and
Center for Brain Science
Harvard University
Cambridge, MA 02138 USA
haim@fiz.huji.ac.il

## Abstract

How are the spatial patterns of spontaneous and evoked population responses related? We study the impact of connectivity on the spatial pattern of fluctuations in the input-generated response, by comparing the distribution of evoked and intrinsically generated activity across the different units of a neural network. We develop a complementary approach to principal component analysis in which separate high-variance directions are derived for each input condition. We analyze subspace angles to compute the difference between the shapes of trajectories corresponding to different network states, and the orientation of the low-dimensional subspaces that driven trajectories occupy within the full space of neuronal activity. In addition to revealing how the spatiotemporal structure of spontaneous activity affects input-evoked responses, these methods can be used to infer input selectivity induced by network dynamics from experimentally accessible measures of spontaneous activity (e.g. from voltage- or calcium-sensitive optical imaging experiments). We conclude that the absence of a detailed spatial map of afferent inputs and cortical connectivity does not limit our ability to design spatially extended stimuli that evoke strong responses.

# 1   Motivation

Stimulus selectivity in neural networks was historically measured directly from input-driven responses [1], and only later were similar selectivity patterns observed in spontaneous activity across the cortical surface [2, 3]. We argue that it is possible to work in the reverse order, and show that analyzing the distribution of spontaneous activity across the different units in the network can inform us about the selectivity of evoked responses to stimulus features, even when no apparent sensory map exists.

Sensory-evoked responses are typically divided into a signal component generated by the stimulus and a noise component corresponding to ongoing activity that is not directly related to the stimulus. Subsequent effort focuses on understanding how the signal depends on properties of the stimulus, while the remaining, irregular part of the response is treated as additive noise. The distinction between external stochastic processes and the noise generated deterministically as a function of intrinsic recurrence has been previously studied in chaotic neural networks [4]. It has also been suggested that internally generated noise is not additive and can be more sensitive to the frequency and amplitude of the input, compared to the signal component of the response [5 - 8].

In this paper, we demonstrate that the interaction between deterministic intrinsic noise and the spatial properties of the external stimulus is also complex and nonlinear. We study the impact of network connectivity on the spatial pattern of input-driven responses by comparing the structure of evoked and spontaneous activity, and show how the unique signature of these dynamics determines the selectivity of networks to spatial features of the stimuli driving them.

# 2   Model description

In this section, we describe the network model and the methods we use to analyze its dynamics. Subsequent sections explore how the spatial patterns of spontaneous and evoked responses are related in terms of the distribution of the activity across the network. Finally, we show how the stimulus selectivity of the network can be inferred from its spontaneous activity patterns.

## 2.1   Network elements

We build a firing rate model of $N$ interconnected units characterized by a statistical description of the underlying circuitry (as $N \to \infty$, the system "self averages" making the description independent of a specific network architecture, see also [11, 12]). Each unit is characterized by an activation variable $x_i \ \forall \ i = 1, 2, \ldots N$, and a nonlinear response function $r_i$ which relates to $x_i$ through $r_i = R_0 + \phi(x_i)$ where,

$$\phi(x) = \begin{cases} R_0 \tanh\left(\frac{x}{R_0}\right) & \text{for } x \leq 0 \\ (R_{\max} - R_0) \tanh\left(\frac{x}{R_{\max} - R_0}\right) & \text{otherwise.} \end{cases} \tag{1}$$

Eq. 1 allows us to independently set the maximum firing rate $R_{\max}$ and the background rate $R_0$ to biologically reasonable values, while retaining a maximum gradient at $x = 0$ to guarantee the smoothness of the transition to chaos [4].

We introduce a recurrent weight matrix with element $J_{ij}$ equivalent to the strength of the synapse from unit $j \to$ unit $i$. The individual weights are chosen independently and randomly from a Gaussian distribution with mean and variance given by $[J_{ij}]_J = 0$ and $[J_{ij}^2]_J = g^2/N$, where square brackets are ensemble averages [9 - 11,13]. The control parameter $g$ which scales as the variance of the synaptic weights, is particularly important in determining whether or not the network produces spontaneous activity with non-trivial dynamics (Specifically, $g = 0$ corresponds to a completely uncoupled network and a network with $g = 1$ generates non-trivial spontaneous activity [4, 9, 10]).

The activation variable for each unit $x_i$ is therefore determined by the relation,

$$\tau_r \frac{dx_i}{dt} = -x_i + g \sum_{j=1}^{N} J_{ij} r_j + I_i \,, \tag{2}$$

with the time scale of the network set by the single-neuron time constant $\tau_r$ of 10 ms.

The amplitude $I$ of an oscillatory external input of frequency $f$, is always the same for each unit, but in some examples shown in this paper, we introduce a neuron-specific phase factor $\theta_i$, chosen randomly from a uniform distribution between $0$ and $2\pi$, such that

$$I_i = I\cos(2\pi ft + \theta_i) \ \ \forall \ \ i = 1, 2, \ldots N. \tag{3}$$

In visually responsive neurons, this mimics a population of simple cells driven by a drifting grating of temporal frequency $f$, with the different phases arising from offsets in spatial receptive field locations. The randomly assigned phases in our model ensure that the spatial pattern of input is not correlated with the pattern of recurrent connectivity. In our selectivity analysis however (Fig. 3), we replace the random phases with spatial input patterns that are aligned with network connectivity.

## 2.2 PCA redux

Principal component analysis (PCA) has been applied profitably to neuronal recordings (see for example [14]) but these analyses often plot activity trajectories corresponding to different network states using the fixed principal component coordinates derived from combined activities under all stimulus conditions. Our analysis offers a complementary approach whereby separate principal components are derived for each stimulus condition, and the resulting principal angles reveal not only the difference between the shapes of trajectories corresponding to different network states, but also the orientation of the low-dimensional subspaces these trajectories occupy within the full $N$-dimensional space of neuronal activity.

The instantaneous network state can be described by a point in an $N$-dimensional space with coordinates equal to the firing rates of the $N$ units. Over time, the network activity traverses a trajectory in this $N$-dimensional space and PCA can be used to delineate the subspace in which this trajectory lies. The analysis is done by diagonalizing the equal-time cross-correlation matrix of network firing rates given by,

$$D_{ij} = \langle (r_i(t) - \langle r_i \rangle)(r_j(t) - \langle r_j \rangle) \rangle \, , \tag{4}$$

where $<>$ denotes a time average. The eigenvalues of this matrix expressed as a fraction of their sum (denoted by $\tilde{\lambda}_a$ in this paper), indicate the distribution of variances across the different orthogonal directions in the activity trajectory.

Spontaneous activity is a useful indicator of recurrent effects, because it is completely determined by network feedback. We can therefore study the impact of network connectivity on the spatial pattern of input-driven responses by comparing the spatial structure of evoked and spontaneous activity. In the spontaneous state, there are a number of significant contributors to the total variance. For instance, for $g = 1.5$, the leading 10% of the components account for 90% of the total variance with an exponential taper for the variance associated with higher components. In addition, projections of network activity onto components with smaller variances fluctuate at progressively higher frequencies, as illustrated in Fig. 1b & d.

Other models of chaotic networks have shown a regime in which an input generates a non-chaotic network response, even though the network returns to chaotic fluctuations when the external drive is turned off [5, 16]. Although chaotic intrinsic activity can be completely suppressed by the input in this network state, its imprint can still be detected in the spatial pattern of the non-chaotic activity. We determine that the perfectly entrained driven state is approximately two-dimensional corresponding to a circular oscillatory orbit, the projections of which are oscillations $\pi/2$ apart in phase. (The residual variance in the higher dimensions reflects harmonics arising naturally from the nonlinearity in the network model).

## 2.3 Dimensionality of spontaneous and evoked activity

To quantify the dimension of the subspace containing the chaotic trajectory in more detail, we introduce the quantity

$$N_{\text{eff}} = \left( \sum_{a=1}^{N} \tilde{\lambda}_a^2 \right)^{-1} , \tag{5}$$

which provides a measure of the effective number of principal components describing a trajectory. For example, if $n$ principal components share the total variance equally, and the remaining $N - n$ principal components have zero variance, $N_{\text{eff}} = n$.

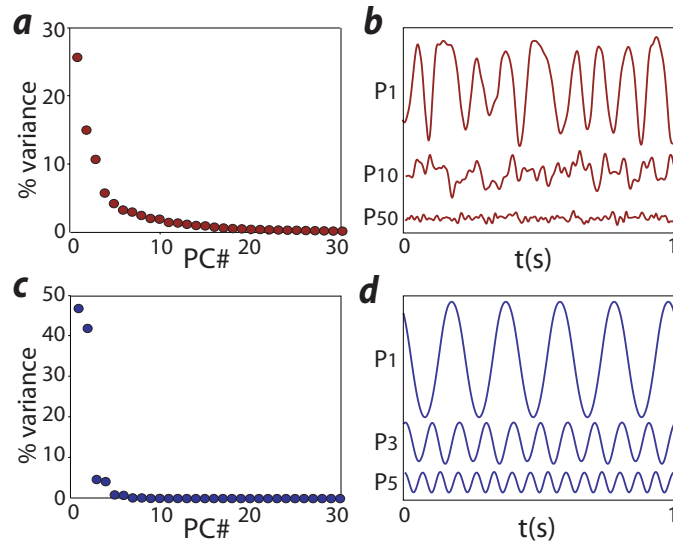

Figure 1: PCA of the chaotic spontaneous state and non-chaotic driven state reached when an input of sufficiently high amplitude has suppressed the chaotic fluctuations. a) % variance accounted for by different PC's for chaotic spontaneous activity. b) Projections of the chaotic spontaneous activity onto PC vectors 1, 10 and 50 (in decreasing order of variance). c) Same as panel a, but for non-chaotic driven activity. d) Projections of periodic driven activity onto PC's 1, 3, and 5. Projections onto components 2, 4, and 6 are identical but phase shifted by $\pi/2$. For this figure, $N = 1000$, $g = 1.5$, $f = 5$ Hz and $I/I_{1/2} = 0.7$ for b and d.

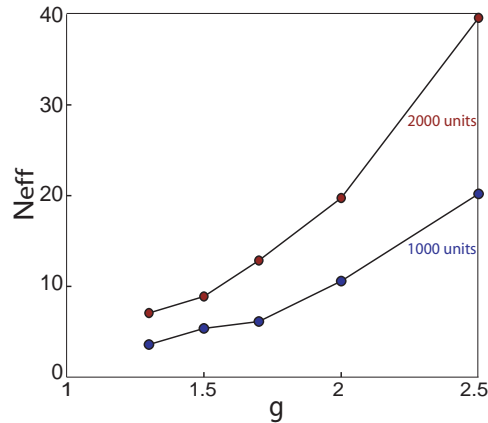

Figure 2: The effective dimension $N_{\text{eff}}$ of the trajectory of chaotic spontaneous activity as a function of $g$ for networks with 1000 (blue circles) or 2000 (red circles) neurons.

For the chaotic spontaneous state in the networks we build, $N_{\text{eff}}$ increases with $g$ (Fig. 2), due to the higher amplitude and frequency content of chaotic dynamics for large $g$. Note that $N_{\text{eff}}$ scales approximately with $N$, which means that large networks have proportionally higher-dimensional chaotic activity (compare the two traces within Fig. 2). The fact that the number of activated modes is only 2% of the system's total dimensionality even for $g$ as high as 2.5, is another manifestation of the deterministic nature of the autonomous fluctuations. For comparison, we calculated $N_{\text{eff}}$ for a similar network driven by white noise, with $g$ set below the chaotic transition at $g = 1$. In this case, $N_{\text{eff}}$ only assumes such low values when $g$ is within a few percent of the critical value of 1.

## 2.4 Subspace angles

The orbit describing the activity in the non-chaotic driven state consists of a circle in a two-dimensional subspace of the full N-dimensions of neuronal activities. How does this circle align relative to the subspaces defined by different numbers of principal components that characterize the spontaneous activity? To overcome the difficulty in visualizing this relationship due to the high dimensionality of both the chaotic subspace as well as the full space of network activities, we utilize principal angles between subspaces [15].

The first principal angle is the angle between two unit vectors (called principal vectors), one in each subspace, that have the maximum overlap (dot product). Higher principal angles are defined recursively as the angles between pairs of unit vectors with the highest overlap that are orthogonal to the previously defined principal vectors. For two subspaces of dimension $d_1$ and $d_2$ defined by the orthogonal unit vectors $V_1^a$, for $a = 1, 2, \ldots d_1$ and $V_2^b$, for $b = 1, 2, \ldots d_2$, the cosines of the principal angles are equal to the singular values of the $d_1 \times d_2$ matrix $V_1^a \cdot V_2^b$. The angle between the two subspaces is given by,

$$\theta = \arccos\left(\min(\mathrm{singular\,value\,of\,} V_1^a \cdot V_2^b)\right) . \tag{6}$$

The resulting principal angles vary between 0 and $\pi/2$ depending on whether the two subspaces overlap partially or whether the two subspaces are completely non-overlapping, respectively. The angle between two subspaces is, by convention, the largest of their principal angles.

## 2.5 Signal and noise from network responses

To characterize the activity of the entire network, we compute the average autocorrelation function of each neuronal firing rate averaged across all the network units, defined as,

$$C(\tau) = \frac{1}{N} \sum_{i=1}^{N} \langle (r_i(t) - \langle r_i \rangle)(r_i(t + \tau) - \langle r_i \rangle) \rangle . \tag{7}$$

The total variance in the fluctuations of the firing rates of the network neurons is denoted by $C(0)$, whereas $C(\tau)$ for non-zero $\tau$ provides information about the temporal structure of the network activity. To quantify signal and noise from this measure of network activity, we split the total variance of the network activity (i.e., $C(0)$) into oscillatory and chaotic components,

$$C(0) = \sigma_{\mathrm{chaos}}^2 + \sigma_{\mathrm{osc}}^2 . \tag{8}$$

As depicted in the function plotted in Fig. 4a, $\sigma_{\mathrm{osc}}^2$ is defined as the amplitude of the oscillatory part of the correlation function $C(\tau)$. The chaotic variance $\sigma_{\mathrm{chaos}}^2$, is then equal to the difference between the full variance $C(0)$ and the variance $\sigma_{\mathrm{osc}}^2$ induced by entrainment to the periodic drive. We call $\sigma_{\mathrm{osc}}$ the signal amplitude and $\sigma_{\mathrm{chaos}}$ the noise amplitude, although it should be kept in mind that this "noise" is generated by the network in a deterministic not stochastic manner [5 - 8].

# 3 Network effects on the spatial pattern of evoked activity

A mean-field-based study developed for chaotic neural networks has recently shown a phase transition in which chaotic background can be actively suppressed by inputs in a temporal frequency-dependent manner [5 - 8]. Similar effects have also been shown in discrete-time models and models with white noise inputs [16, 17] but these models lack the rich dynamics of continuous time models. In contrast, we show that external inputs do not exert nearly as strong control on the spatial structure of the network response. The phases of the firing-rate oscillations of network neurons are only partially correlated with the phases of the inputs that drive them, instead appearing more strongly influenced by the recurrent feedback.

We schematize the irregular trajectory of the chaotic spontaneous activity, described by its leading principal components in red in Fig. 3a. The circular orbit of the periodic activity (schematically in blue in 3a) has been rotated by the smaller of its two principal angles. The angle between these two subspaces (the angle between $\hat{n}_{\mathrm{chaos}}$ and $\hat{n}_{\mathrm{osc}}$) is then the remaining angle through which the periodic orbit would have to be rotated to align it with the horizontal plane containing the two-dimensional projection of the chaotic trajectory. In other words, Fig. 3a depicts the angle between

the subspaces defined by the first two principal components of the orbit of periodic driven activity and the first two principal components of the chaotic spontaneous activity. We ask how this circle is aligned relative to the subspaces defined by different numbers of principal components that characterize the spontaneous activity.

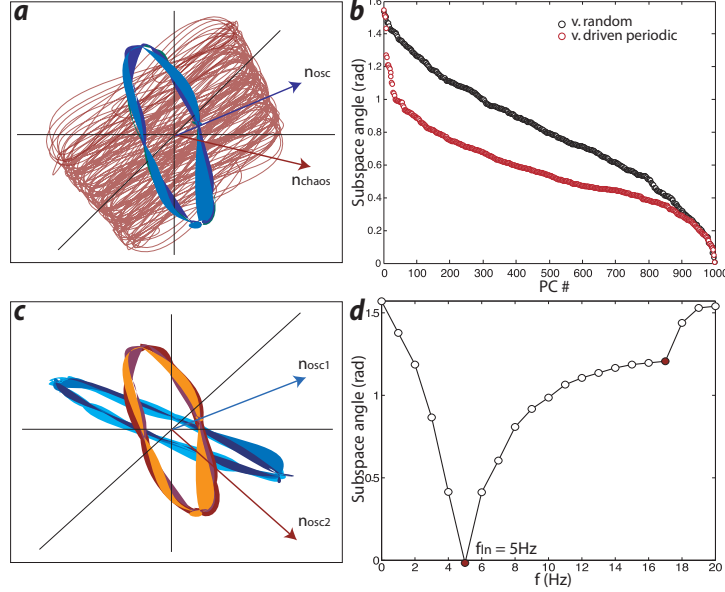

Figure 3: Spatial pattern of network responses. a) Cartoon of the angle between the subspace defined by the first two components of the chaotic activity (red) and a two-dimensional description of the periodic orbit (blue curve). b) Relationship between the orientation of periodic and chaotic trajectories. Angles between the subspace defined by the two PC's of the non-chaotic driven state and subspaces formed by PC's 1 through $m$ of the chaotic spontaneous activity, where $m$ appears on the horizontal axis (red dots). Black dots show the analogous angles but with the two-dimensional subspace defined by the random input phases replacing the subspace of the non-chaotic driven activity. c) Cartoon of the angle between the subspaces defined by two periodic driven trajectories. d) Effect of input frequency on the orientation of the periodic orbit. Angle between the subspaces defined by the two leading principal components of non-chaotic driven activity at different frequencies and these two vectors for a 5 Hz input frequency. The results in this figure come from a network simulation with $N = 1000$ and $I/I_{1/2} = 0.7$ and $f = 5$ Hz for b, $I/I_{1/2} = 1.0$ for d.

Next, we compare the two-dimensional subspace of the periodic driven orbit to the subspaces defined by the first $m$ principal components of the chaotic spontaneous activity. This allows us to see how the orbit lies in the full $N$-dimensional space of neuronal activities relative to the trajectory of the chaotic spontaneous activity. The results (Fig. 3b, red dots) show that this angle is close to $\pi/2$ for small $m$, equivalent to the angle between two randomly chosen subspaces. However, the value drops quickly for subspaces defined by progressively more of the leading principal components of the chaotic activity. Ultimately, this angle approaches zero when all $N$ of the chaotic principal component vectors are considered, as it must, because these span the entire space of network activities.

In the periodic driven regime, the temporal phases of the different neurons determine the orientation of the orbit in the space of neuronal activities. The rapidly falling angle between this orbit and the subspaces defined by spatial patterns dominating the chaotic state (Fig. 3b, red dots) indicates that these phases are strongly influenced by the recurrent connectivity, that in turn determines the spatial pattern of the spontaneous activity. As an indication of the magnitude of this effect, we note that the angles between the random phase sinusoidal trajectory of the input and the same chaotic subspaces are much larger than those associated with the periodic driven activity (Fig. 3b, black dots).

# 4 Temporal frequency modulation of spatial patterns

Although recurrent feedback in the network plays an important role in the structure of driven network responses, the spatial pattern of the activity is not fixed but rather, is shaped by a complex interaction between the driving input and intrinsic network dynamics. It is therefore sensitive to both the amplitude and the frequency of this drive. To see this, we examine how the orientation of the approximately two-dimensional periodic orbit of driven network activity in the non-chaotic regime depends on input frequency. We use the technique of principal angles described above, to examine how the orientation of the oscillatory orbit changes when the input frequency is varied (angle between $\hat{n}_{\mathrm{osc1}}$ and $\hat{n}_{\mathrm{osc2}}$ in Fig. 3c). For comparison purposes, we choose the dominant two-dimensional subspace of the network oscillatory responses to a driving input at 5 Hz as a reference. We then calculate the principal angles between this subspace and the corresponding subspaces evoked by inputs with different frequencies. The result shown in Fig. 3d indicates that the orientation of the orbit for these driven states rotates as the input frequency changes.

The frequency dependence of the orientation of the evoked response is likely related to the effect seen in Fig. 1b & d in which higher frequency activity is projected onto higher principal components of the spontaneous activity. This causes the orbit of the driven activity to rotate in the direction of higher-order principal components of the spontaneous activity as the input frequency increases. In addition, we find that the larger the stimulus amplitude, the closer the response phases of the neurons are to the random phases of their external inputs (results not shown).

# 5 Network selectivity

We have shown that the response of a network to random-phase input is strongly affected by the spatial structure of spontaneous activity (Fig. 3b). We now ask if the spatial patterns that dominate the spontaneous activity in a network correspond to the spatial input patterns to which the network responds most robustly. In other words, can the spatial structure of an input be designed to maximize its ability to suppress chaos?

Rather than using random-phase inputs, we align the inputs to our network along the directions defined by the different principal components of its spontaneous activity. Specifically, the input to neuron $i$ is set to,

$$I_i = I V_i^a \cos(2\pi f t) , \qquad (9)$$

where $I$ is the amplitude factor and $V_i^a$ is the $i^{\mathrm{th}}$ component of principal component vector $a$ of the spontaneous activity. The index $a$ is ordered so that $a = 1$ corresponds to the principal component with the largest variance and $a = N$, the least.

The signal amplitude when the input is aligned with different leading eigenvectors shows no strong dependence on $a$, but the noise amplitude exhibits a sharp transition from no chaotic component for small $a$ to partial chaos for larger $a$ (Fig.4b). The critical value of $a$ depends on $I$, $f$ and $g$ but, in general, inputs aligned with the directions along which the spontaneous network activity has large projections are most effective at inducing transitions to the driven periodic state. The point $a = 5$ corresponds to a phase transition analogous to that seen in other network models [5, 16]. The noise is therefore more sensitive to the spatial structure of the input compared to the signal.

Suppression of spontaneously generated noise in neural networks does not require stimuli so strong that they simply overwhelm fluctuations through saturation. Near the onset of chaos, complete noise suppression can be achieved with relatively low amplitude inputs (compared to the strength of the internal feedback), especially if the input is aligned with the dominant principal components of the spontaneous activity.

# 6 Discussion

Many models of selectivity in cortical circuits rely on knowledge of the spatial organization of afferent inputs as well as cortical connectivity. However, in many cortical areas, such information is not available. This is analogous to the random character of connectivity in our network which precludes

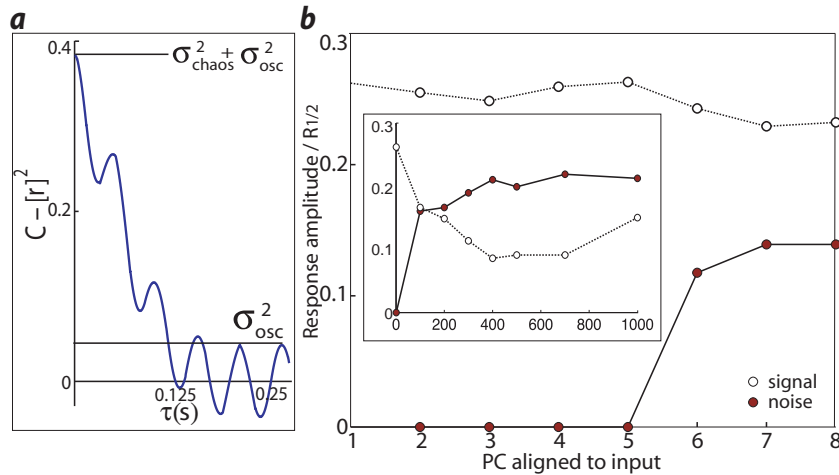

Figure 4: a) An example autocorrelation function. Horizontal lines indicate how we define the signal and noise amplitudes. Parameters used for this figure are $I/I_{1/2} = 0.4$, $g = 1.8$ and $f = 20$ Hz. b) Network selectivity to different spatial patterns of input. Signal and noise amplitudes in the input-evoked response aligned to the leading principal components of the spontaneous activity of the network. The inset shows a larger range on a coarser scale. The results in this figure come from a network simulation with $N = 1000$, $I/I_{1/2} = 0.2$ and $f = 2$ Hz for b.

a simple description of the spatial distribution of activity patterns in terms of topographically organized maps. Our analysis shows that even in cortical areas where the underlying connectivity does not exhibit systematic topography, dissecting the spatial patterns of fluctuations in neuronal activity can yield important insight about both intrinsic network dynamics and stimulus selectivity.

Analysis of the spatial pattern of network activity reveals that even though the network connectivity matrix is full rank, the effective dimensionality of the chaotic fluctuations is much smaller than network size. This suppression of spatial modes is much stronger than expected, for instance, from a linear network that low-pass filters a spatiotemporal white noise input. Further, this study extends a similar effect demonstrated in the temporal domain elsewhere [5 - 8] to show that active spatial patterns exhibit strong nonlinear interaction between external driving inputs and intrinsic dynamics. Surprisingly though, even when the input is strong enough to fully entrain the temporal pattern of network activity, spatial organization of the activity remains strongly influenced by recurrent dynamics.

Our results show that experimentally accessible spatial patterns of spontaneous activity (e.g. from voltage- or calcium-sensitive optical imaging experiments) can be used to infer the stimulus selectivity induced by the network dynamics and to design spatially extended stimuli that evoke strong responses. This is particularly true when selectivity is measured in terms of the ability of a stimulus to entrain the neural dynamics. In general, our results indicate that the analysis of spontaneous activity can provide valuable information about the computational implications of neuronal circuitry.

**Acknowledgments**

Research of KR and LFA supported by National Science Foundation grant IBN-0235463 and an NIH Director's Pioneer Award, part of the NIH Roadmap for Medical Research, through grant number 5-DP1-OD114-02. HS was partially supported by grants from the Israel Science Foundation and the McDonnell Foundation. This research was also supported by the Swartz Foundation through the Swartz Centers at Columbia, Princeton and Harvard Universities.

**References**

[1] Hubel, D.H. & Wiesel, T.N. (1962) *Receptive fields, binocular interaction and functional architecture in the cats visual cortex.* J. Physiol. 160, 106-154.

[2] Arieli, A., Shoham, D., Hildesheim, R. & Grinvald, A. (1995) *Coherent spatiotemporal patterns of ongoing activity revealed by real-time optical imaging coupled with single-unit recording in the cat visual cortex.* J. Neurophysiol. 73, 2072-2093.

[3] Arieli, A., Sterkin, A., Grinvald, A. & Aertsen, A. (1996) *Dynamics of ongoing activity: explanation of the large variability in evoked cortical responses.* Science 273, 1868-1871.

[4] Sompolinsky, H., Crisanti, A. & Sommers, H.J. (1988) *Chaos in Random Neural Networks.* Phys. Rev. Lett. 61, 259-262.

[5] Rajan, K., Abbott, L.F. & Sompolinsky, H. (2010) *Stimulus-dependent Suppression of Chaos in Recurrent Neural Networks.* Phys. Rev. E., 82: 01193.

[6] Rajan, K. (2009) *Nonchaotic Responses from Randomly Connected Networks of Model Neurons.* Ph.D. Dissertation, Columbia University in the City of New York.

[7] Rajan, K., Abbott, L. F., & Sompolinsky, H. (2010) *Stimulus-dependent Suppression of Intrinsic Variability in Recurrent Neural Networks.* BMC Neuroscience, 11, O17: 11.

[8] Rajan, K. (2010) *What do Random Matrices Tell us about the Brain?* Grace Hopper Celebration of Women in Computing, published by the Anita Borg Institute for Women & Technology and the Association for Computing Machinery.

[9] van Vreeswijk, C. & Sompolinsky, H. (1996) *Chaos in neuronal networks with balanced excitatory and inhibitory activity.* Science 24, 1724-1726.

[10] van Vreeswijk, C. & Sompolinsky, H. (1998) *Chaotic balanced state in a model of cortical circuits.* Neural Comput. 10, 1321-1371.

[11] Shriki, O., Hansel, D. & Sompolinsky, H. (2003) *Rate models for conductance-based cortical neuronal networks.* Neural Comput. 15, 1809-1841.

[12] Wong, K.-F. & Wang, X.-J. (2006) *A Recurrent network mechanism of time integration in perceptual decisions.* J. Neurosci. 26, 1314-1328.

[13] Rajan, K. & Abbott, L.F. (2006) *Eigenvalue spectra of random matrices for neural networks.* Phys. Rev. Lett. 97, 188104.

[14] Broome, B.M., Jayaraman, V. & Laurent, G. (2006) *Encoding and decoding of overlapping odor sequences.* Neuron 51, 467-482.

[15] Ipsen, I.C.F. & Meyer, C.D. (1995) *The angle between complementary subspaces.* Amer. Math. Monthly 102, 904-911.

[16] Bertchinger, N. & NatschŁger, T. (1995) *Real-time computation at the edge of chaos in recurrent neural networks.* Neural Comput. 16, 1413-1436.

[17] Molgedey, L., Schuchhardt, J. & Schuster, H.G. (1992) *Suppressing chaos in neural networks by noise.* Phys. Rev. Lett. 69, 3717-3719.

